# Blind Separation of Radio Signals in Fading Channels

**Kari Torkkola**

Motorola, Phoenix Corporate Research Labs,
2100 E. Elliot Rd, MD EL508, Tempe, AZ 85284, USA
email: `A540AA@email.mot.com`

## Abstract

We apply information maximization / maximum likelihood blind source separation [2, 6] to complex valued signals mixed with complex valued nonstationary matrices. This case arises in radio communications with baseband signals. We incorporate known source signal distributions in the adaptation, thus making the algorithms less "blind". This results in drastic reduction of the amount of data needed for successful convergence. Adaptation to rapidly changing signal mixing conditions, such as to fading in mobile communications, becomes now feasible as demonstrated by simulations.

## 1 Introduction

In SDMA (spatial division multiple access) the purpose is to separate radio signals of interfering users (either intentional or accidental) from each others on the basis of the spatial characteristics of the signals using smart antennas, array processing, and beamforming [5, 8]. Supervised methods typically use a variant of LMS (least mean squares), either gradient based, or algebraic, to adapt the coefficients that describe the channels or their inverses. This is usually a robust way of estimating the channel but a part of the signal is wasted as predetermined training data, and the methods might not be fast enough for rapidly varying fading channels.

Unsupervised methods either rely on information about the antenna array manifold, or properties of the signals. Former approaches might require calibrated antenna arrays or special array geometries. Less restrictive methods use signal properties only, such as constant modulus, finite alphabet, spectral self-coherence, or cyclostationarity. Blind source separation (BSS) techniques typically rely only on source signal independence and non-Gaussianity assumptions.

Our aim is to separate simultaneous radio signals occupying the same frequency band, more specifically, radio signals that carry digital information. Since linear mixtures of antenna signals end up being linear mixtures of (complex) baseband signals due to the linearity of the downconversion process, we will apply BSS at the baseband stage of the receiver. The main contribution of this paper is to show that by making better use of the known signal properties, it is possible to devise algorithms that adapt much faster than algorithms that rely only on weak assumptions, such as source signal independence.

We will first discuss how the probability density functions (pdf) of baseband DPSK signals could be modelled in a way that can efficiently be used in blind separation algorithms. We will incorporate those models into information maximization and

into maximum likelihood approaches [2, 6]. We will then continue with the maximum likelihood approach and other modulation techniques, such as QAM. Finally, we will show in simulations, how this approach results in an adaptation process that is fast enough for fading channels.

## 2   Models of baseband signal distributions

In digital communications the binary (or n-ary) information is transmitted as discrete combinations of the amplitude and/or the phase of the carrier signal. After downconversion to baseband the instantaneous amplitude of the carrier can be observed as the length of a complex valued sample of the baseband signal, and the phase of the carrier is discernible as the phase angle of the same sample. Possible combinations that depend on the modulation method employed, are called symbol constellations. N-QAM (quadrature amplitude modulation) utilizes both the amplitude and the phase, whereby the baseband signals can only take one of N possible locations on a grid on the complex plane. In N-PSK (phase shift keying) the amplitude of the baseband signal stays constant, but the phase can take any of N discrete values. In DPSK (differential phase shift keying) the information is encoded as the difference between phases of two consecutive transmitted symbols. The phase can thus take any value, and since the amplitude remains constant, the baseband signal distribution is a circle on the complex plane.

Information maximization BSS requires a nonlinear function that models the cumulative density function (cdf) of the data. This function and its derivative need to be differentiable. In the case of a circular complex distribution with uniformly distributed phase, there is only one important direction of deviation, the radial direction. A smooth cdf $G$ for a circular distribution at the unit circle can be constructed using the hyperbolic tangent function as

$$G(z) = tanh(w(|z| - 1)) \tag{1}$$

and the pdf, differentiated in the radial direction, that is, with respect to $|z|$ is

$$g(z) = \frac{\partial}{\partial|z|}tanh(w(|z| - 1)) = w(1 - tanh^2(w(|z| - 1))) \tag{2}$$

where $z = x + iy$ is a complex valued variable, and the parameter $w$ controls the steepness of the slope of the *tanh* function. Note that this is in contrast to more commonly used coordinate axis directions to differentiate and to integrate to get the pdf from the cdf and vice versa. These functions are plotted in Fig. 1.

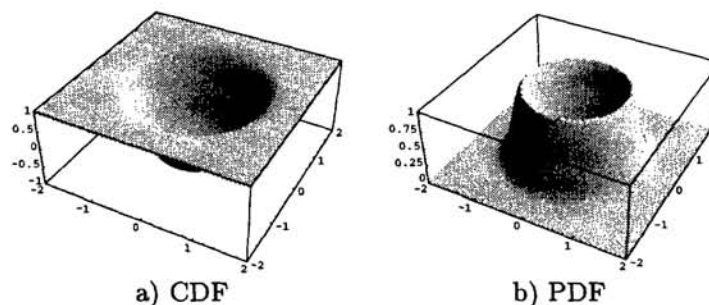

a) CDF          b) PDF

Figure 1: Radial *tanh* with w=2.0 (equations 1 and 2).

Note that we have not been worrying about the pdf integrating to unity. Thus we could leave the first multiplicative constant $w$ out of the definition of $g$. Scaling will not be important for our purposes of using these functions as the nonlinearities in the information maximization BSS. Note also that when the steepness $w$ approaches infinity, the densities approach the ideal density of a DPSK source, the unit circle. Many other equally good choices are possible where the ideal density is reached as a limit of a parameter value. For example, the radial section of the circular "ridge" of the pdf could be a Gaussian.

## 3    The information maximization adaptation equation

The information maximization adaptation equation to learn the unmixing matrix $W$ using the natural gradient is [2]

$$\Delta W \propto (\hat{y}u^T + I)W \quad \text{where} \quad \hat{y}_j = \frac{\partial}{\partial y_j}\frac{\partial y_j}{\partial u_j} \tag{3}$$

Vector $u = Wx$ denotes a time sample of the separated sources, $x$ denotes the corresponding time sample of the observed mixtures, and $y_j$ is the nonlinear function approximating the cdf of the data, which is applied to each component of the $u$.

Now we can insert (1) into $y_j$. Making use of $\partial|z|/\partial z = z/|z|$ this yields for $\hat{y}_j$:

$$\hat{y}_j = \frac{\partial}{\partial y_j}\frac{\partial}{\partial u_j} tanh(w(|u_j| - 1)) = -2wy_j\frac{u_j}{|u_j|} \tag{4}$$

When (4) is inserted into (3) we get

$$\Delta W \propto \left( I - 2\left(\frac{w_j tanh(w_j(|u_j| - 1))u_j}{|u_j|}\right)_j u^H \right) W. \tag{5}$$

where $(.)_j$ denotes a vector with elements of varying $j$. Here, we have replaced the transpose operator by the hermitian operator $H$, since we will be processing complex data. We have also added a subscript to $w$ as these parameters can be learned, too. We will not show the adaptation equations due to lack of space.

## 4    Connection to the maximum likelihood approach

Pearlmutter and Parra have shown that (3) can be derived from the maximum likelihood approach to density estimation [6]. The same fact has also been pointed out by others, for example, by Cardoso [3]. We will not repeat their straightforward derivation, but the final adaptation equation is of the following form:

$$\Delta W \propto -\frac{d\hat{G}}{dW}W^T W = \left( \left(\frac{f_j'(u_j;w_j)}{f_j(u_j;w_j)}\right)_j u^T + I \right) W. \tag{6}$$

where $u = Wx$ are the sources separated from mixtures $x$, and $f_j(u_j;w_j)$ is the pdf of source $j$ parametrized by $w_j$. This is exactly the form of Bell and Sejnowski when $f_j$ is taken to be the derivative of the necessary nonlinearity $g_j$, which was assumed to be "close" to the true cdf of the source. Thus the information maximization approach makes implicit assumptions about the cdf's of the sources in the form of the nonlinear squashing function, and does implicit density estimation, whereas in the ML approach the density assumptions are made explicit. This fact makes it more intuitive and lucid to derive the adaptation for other forms of densities, and also to extend it to complex valued variables.

Now, we can use the circular pdf's (2) depicted in Fig. 1 as the densities $f_j$ (omitting scaling) $f_j(u_j;w_j) = 1 - tanh^2(w_j(|u_j| - 1))$. where the steepness $w_j$ acts as the single parameter of the density. Now we need to compute its derivative

$$f_j'(u_j;w_j) = \frac{\partial}{\partial u_j}f_j(u_j;w_j) = -2\, tanh(w_j(|u_j| - 1))f_j(u_j;w_j)w_j\frac{u_j}{|u_j|} \tag{7}$$

Inserting this into (6) and changing transpose operators into hermitians yields

$$\Delta W \propto \left( I - 2\left(\frac{w_j tanh(w_j(|u_j| - 1))u_j}{|u_j|}\right)_j u^H \right) W, \tag{8}$$

which is exactly the information maximization rule (5). Notice that at this time we did not have to ponder what would be an appropriate way to construct the cdf from the pdf for complex valued distributions.

## 5   Modifications for QAM and other signal constellations

So far we have only looked at signals that lie on the unit circle, or that have a constant modulus. Now we will take a look at other modulation techniques, in which the alphabet is constructed as discrete points on the complex plane. An example is the QAM (quadrature amplitude modulation), in which the signal alphabet is a regular grid. For example, in 4-QAM, the alphabet could be $A_4 = \{1+i, -1+i, -1-i, 1-i\}$, or any scaled version of $A_4$.

In the ideal pdf of 4-QAM, each symbol is represented just as a point. Again, we can construct a smoothed version of the ideal pdf as the sum of "bumps" over all of the alphabet where the ideal pdf will be approached by increasing $w$.

$$g(u) = \sum_k (1 - tanh^2(w_k|u - u_k|)) \tag{9}$$

Now the density for each source $j$ will be

$$f_j(u_j; \mathbf{w}_j) = \sum_k (1 - tanh^2(w_k|u_j - u_k|)) \tag{10}$$

where $\mathbf{w}_j$ is now a vector of parameters $w_k$. In practice each $w_k$ would be equal in which case a single parameter $w$ will suffice.

This density function could now be inserted into (6) resulting in the weight update equation. However, since $f_j(u_j; \mathbf{w}_j)$ is a sum of multiple components, $f'/f$ will not have a particularly simple form. In essence, for each sample to be processed, we would need to evaluate all the components of the pdf model of the constellation. This can be avoided by evaluating only the component of the pdf corresponding to that symbol of the alphabet $u_c$ which is nearest to the current separated sample $u$. This is a very good approximation when $w$ is large. But the approximation does not even have to be a good one when $w$ is small, since the whole purpose of using "wide" pdf components is to be able to evaluate the gradients on the whole complex plane. Figure 2 depicts examples of this approximation with two different values of $w$. The discontinuities are visible at the real and imaginary axes for the smaller $w$.

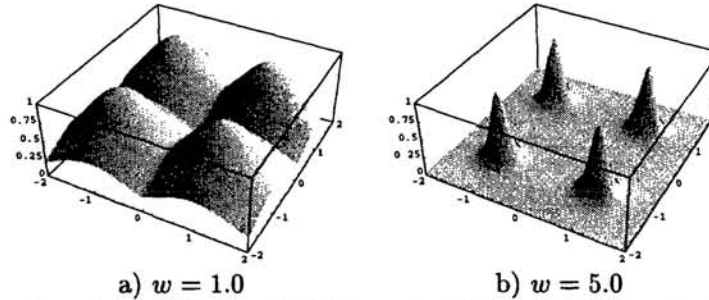

a) $w = 1.0$                    b) $w = 5.0$

Figure 2: A piecewise continuous PDF for a 4-QAM source using the *tanh* function.

Thus for the 4-QAM, the complex plane will be divided into 4 quadrants, each having its own adaptation rule corresponding to the single pdf component in that quadrant. Evaluating (6) for each component of the sum gives

$$\Delta_k W \propto \left(I - 2\left(\frac{w_k tanh(w_k|u_j - u_k|)u_j}{|u_j|}\right)_j u^H\right) W, \tag{11}$$

for each symbol $k$ of the alphabet or for the corresponding location $u_k$ on the complex plane. This equation can be applied as such when the baseband signal is sampled at the symbol rate. With oversampling, it may be necessary to include in the pdf model the transition paths between the symbols, too.

# 6   Practical simplifications

To be able to better vectorize the algorithm, it is practical to accumulate $\Delta W$ from a number of samples before updating the $W$. This amounts to computing an expectation of $\Delta W$ over a number, say, 10-500 samples of the mixtures. Looking at the DPSK case, (5) or (8) the expectation of $|u_i|$ in the denominator equals one "near" convergence since we assume baseband signals that are distributed on the unit circle.

Also, near the solution we can assume that the separated outputs $u_j$ are close to true distributions, the exact unit circle, which can be derived from $f_j$ by increasing its steepness. At the limit the *tanh* will equal the *sign* function, when the whole adaptation, ignoring scaling, is

$$\Delta W \propto \left(I - 2(sign(|u_j| - 1)u_j)_j u^H\right) W, \tag{12}$$

However, this simplification can only be used when the $W$ is not too far off from the correct solution. This is especially true when the number of available samples of the mixtures is small. The smooth *tanh* is needed in the beginning of the adaptation to give the correct direction to the gradient in the algorithm since the pdfs of the outputs $u_j$ are far from the ideal ones in the beginning.

# 7   Performance with static and fading signals

We have tested the performance of the proposed algorithm both with static and dynamic (changing) mixing conditions. In the static case with four DPSK signals (8 x oversampled) mixed with random matrices the algorithm needs only about 80 sample points (corresponding to 10 symbols) of the mixtures to converge to a separating solution, whereas a more general algorithm, such as [4], needs about 800-1200 samples for convergence. We attribute this improvement to making much better use of the baseband signal distributions.

In mobile communications the signals are subject to *fading*. If there is no direct line of sight from the transmitter to the receiver, only multiple reflected and diffracted signal components reach the receiver. When either the receiver or the transmitter is moving, for example, in an urban environment, these components are changing very rapidly. If the phases of the carrier signals in these components are aligned the components add constructively at the receiver. If the phases of carriers are 180 degrees off the components add destructively.

Note that a half of a wavelength difference in the lengths of the paths of the received components corresponds to a 180 degree phase shift. This is only about 0.17 m at 900 MHz. Since this small a spatial difference can cause the signal to change from constructive interference to a null received signal, the result is that both the amplitude and the phase of the received signal vary seemingly randomly at a rate that is proportional to relative speeds of the transmitter and the receiver. The amplitude of the received signal follows a Rayleigh distribution, hence the name *Rayleigh fading*. As an example, Figure 3 depicts a 0.1 second fragment of the amplitude of a fading channel.

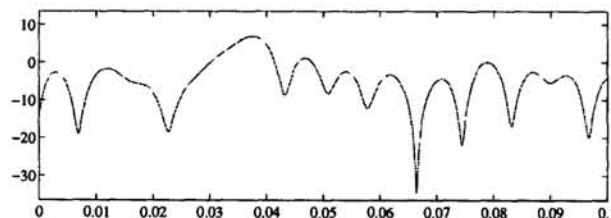

Figure 3: Amplitude (in dB) of a fading radio channel corresponding to a vehicle speed of 60 mph, when the carrier is 900 Mhz. Horizontal axis is time in seconds.

With fading sources, the problem is to be able to adapt to changing conditions, keeping up with the fading rate. In the signal of Fig. 3 it takes less than 5 milliseconds to move from a peak of the amplitude into a deep fade. Assuming a symbol rate of 20000 symbols/second, this corresponds to a mere 100 symbols during this change.

We simulated again DPSK sources oversampling by 8 relative to the symbol rate. The received sampled mixtures are

$$x_i[n] = \sum_j f_{ij}[n]s_j[n] + n_i[n] \tag{13}$$

where $s_j[n]$ are the source signals, $f_{ij}[n]$ represents the fading channel from transmitter $j$ to receiver $i$, and $n_i[n]$ represents the noise observed by receiver $i$.

In our experiments, we used a sliding window of 80 samples centered at the current sample. The weight matrix update (the gradient) was calculated using all the samples of the window, the weight matrix was updated, the window was slid one sample forward, and the same was repeated. Using this technique we were able to keep up with the fading rate corresponding to 60 mph relative speed of the transmitter and the receiver. Figure 4 depicts how the algorithm tracks the fading channels in the case of three simultaneous source signals.

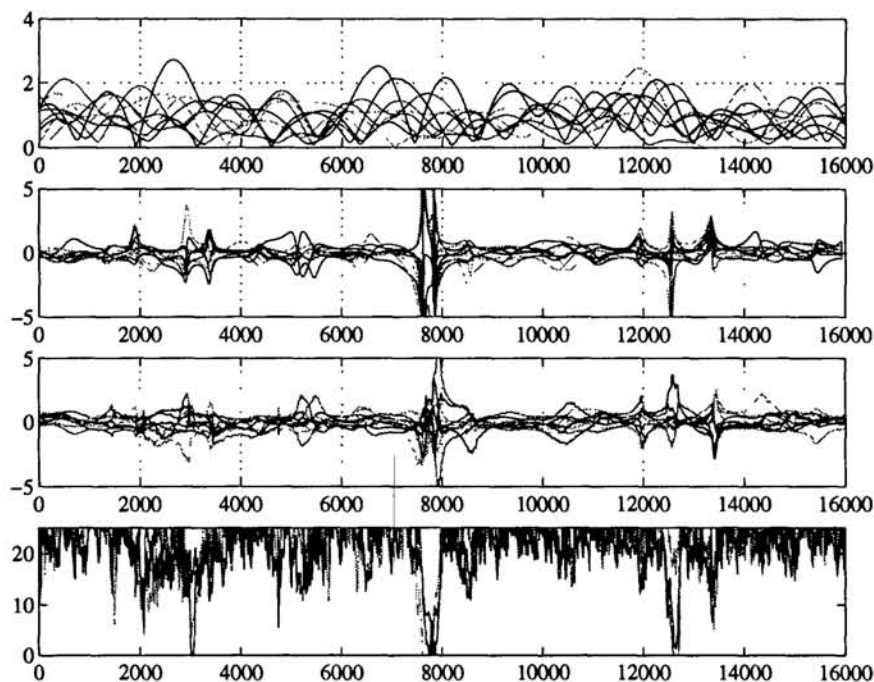

Figure 4: Separation of three signals subject to fading channels. Top graph: The real parts 16 independent fading channels. 2nd graph: The inverse of the instantaneous fading conditions (only the real part is depicted). This is one example of an ideal separation solution. 3rd graph: The separation solution tracked by the algorithm. (only the real part is depicted). Bottom graph: The resulting signal/interference (S/I) ratio in dB for each of the four separated source signals. Horizontal axis is samples. 16000 samples (8 x oversampled) corresponds to 0.1 seconds.

On the average, the S/I to start with is zero. The average output S/I is 20 dB for the worst of the three separated signals. Since the mixing is now dynamic the instantaneous mixing matrix, as determined by the instantaneous fades, can occasionally be singular and cannot be inverted. Thus the signals at this instance cannot be separated. In our 0.1 second test signal this occurred four times in the three source signal case (9 independent fading paths), at which instances the output

S/I bounced to or near zero momentarily for one or more of the separated signals. Durations of these instances are short, lasting about 15 symbols, and covering about 3 per cent of the total signal time.

## 8   Related work and discussion

Although the whole field of blind source separation has started around 1985, rather surprisingly, no application to radio communications has yet emerged. Most of the source separation algorithms are based on higher-order statistics, and these should be relatively straightforward to generalize for complex valued baseband data. Perhaps the main reason is that all theoretical work has concentrated in the case of static mixing, not in the dynamic case. Many communications channels are dynamic in nature, and thus rapidly adapting methods are necessary.

Making use of all available knowledge of the sources, in this case the pdf's of the source signals, allows successful adaptation based on a very small number of samples, much smaller than by just incorporating the coarse shapes of the pdf's into the algorithm. It is not unreasonable to presume this knowledge, on the contrary, the modulation method of a communications system must certainly be known. To our knowledge, no successful blind separation of signals subject to rapidly varying mixing conditions, such as fading, has been reported in the literature.

Different techniques applied to separation of various simulated radio signals under static mixing conditions have been described, for example, in [9, 4]. The maximum likelihood method reported recently by Yellin and Friedlander [9] seems to be the closest to our approach, but they only apply it to simulated baseband radio signals with static mixing conditions.

It must also be noted that channel time dispersion is not taken into account in our current simulations. This is valid only in cases where the delay spread is short compared to the inverse of the signal bandwidths. If this is not a valid assumption, separation techniques for convolutive mixtures, such as in [7] or [1], need to be combined with the methods developed in this paper.

## References

[1] S. Amari, S. Douglas, A. Cichocki, and H. H. Yang. Multichannel blind deconvolution and equalization using the natural gradient. In *Proc. 1st IEEE Signal Processing Workshop on Signal Processing Advances in Wireless Communications*, pages 101–104, Paris, France, April 16-18 1997.

[2] A. Bell and T. Sejnowski. An information-maximisation approach to blind separation and blind deconvolution. *Neural Computation*, 7(6):1129–1159, 1995.

[3] J.-F. Cardoso. Infomax and maximum likelihood for source separation. *IEEE Letters on Signal Processing*, 4(4):112–114, April 1997.

[4] J.-F. Cardoso and B. Laheld. Equivariant adaptive source separation. *IEEE Transactions on Signal Processing*, 44(12):3017–3030, December 1996.

[5] A. Paulraj and C. B. Papadias. Array processing in mobile communications. In *Handbook of Signal Processing*. CRC Press, 1997.

[6] B. A. Pearlmutter and L. C. Parra. A context-sensitive generalization of ICA. In *International Conference on Neural Information Processing*, Hong Kong, Sept. 24–27 1996. Springer.

[7] K. Torkkola. Blind separation of convolved sources based on information maximization. In *IEEE Workshop on Neural Networks for Signal Processing*, pages 423–432, Kyoto, Japan, September 4-6 1996.

[8] A.-J. van der Veen and A. Paulraj. An analytical constant modulus algorithm. *IEEE Transactions on Signal Processing*, 44(5), May 1996.

[9] D. Yellin and B. Friedlander. A maximum likelihood approach to blind separation of narrowband digital communication signals. In *Proc. 30th Asilomar Conf. on Signals, Systems, and Computers*, 1996.
